# Competition Among Networks Improves Committee Performance

**Paul W. Munro**
Department of Information Science
and Telecommunications
University of Pittsburgh
Pittsburgh PA   15260
*munro@sis.pitt.edu*

**Bambang Parmanto**
Department of Health Information
Management
University of Pittsburgh
Pittsburgh PA   15260
*parmanto+@pitt.edu*

## ABSTRACT

The separation of generalization error into two types, bias and variance (Geman, Bienenstock, Doursat, 1992), leads to the notion of error reduction by averaging over a "committee" of classifiers (Perrone, 1993). Committee performance decreases with both the average error of the constituent classifiers and increases with the degree to which the misclassifications are correlated across the committee. Here, a method for reducing correlations is introduced, that uses a winner-take-all procedure similar to competitive learning to drive the individual networks to different minima in weight space with respect to the training set, such that correlations in generalization performance will be reduced, thereby reducing committee error.

## 1   INTRODUCTION

The problem of constructing a predictor can generally be viewed as finding the right combination of bias and variance (Geman, Bienenstock, Doursat, 1992) to reduce the expected error. Since a neural network predictor inherently has an excessive number of parameters, reducing the prediction error is usually done by reducing variance. Methods for reducing neural network complexity can be viewed as a regularization technique to reduce this variance. Examples of such methods are Optimal Brain Damage (Le Cun et. al., 1991), weight decay (Chauvin, 1989), and early stopping (Morgan & Boulard, 1990).

The idea of combining several predictors to form a single, better predictor (Bates & Granger, 1969) has been applied using neural networks in recent years (Wolpert, 1992; Perrone, 1993; Hashem, 1994).

## 2 REDUCING MISCLASSIFICATION CORRELATION

Since committee errors occur when too many individual predictors are in error, committee performance improves as the correlation of network misclassifications decreases. Error correlations can be handled by using a weighted sum to generate a committee prediction; the weights can be estimated by using ordinary least squares (OLS) estimators (Hashem, 1994) or by using Lagrange multipliers (Perrone, 1993).

Another approach (Parmanto et al., 1994) is to reduce error correlation directly by attempting to drive the networks to different minima in weight space, that will presumably have different generalization *syndromes*, or patterns of error with respect to a test set (or better yet, the entire stimulus space).

### 2.1 Data Manipulations

Training the networks using nonidentical data has been shown to improve committee performance, both when the data sets are from mutually exclusive continuous regions (eg, Jacobs et al.,1991), or when the training subsets are arbitrarily chosen (Breiman, 1992; Parmanto, Munro, and Doyle, 1995). Networks tend to converge to different weight states, because the error surface itself depends on the training set; hence changing the data changes the error surface.

### 2.2 Auxiliary tasks

Another way to influence the networks to disagree is to introduce a second output unit with a different task to each network in the committee. Thus, each network has two outputs, a *primary unit* which is trained to predict the class of the input, and a *secondary unit*, with some other task that is *different* than the tasks assigned to the secondary units of the other committee members. *The success of this approach rests on the assumption that the decorrelation of the network errors will more than compensate for any degradation of performance induced on the primary task by the auxiliary task.* The presence of a hidden layer in each network guarantees that the two output response functions share some weight parameters (i.e., the input-hidden weights), and so the learning of the secondary task influences the function learned by the primary output unit.

Parmanto et al. (1994) acheived significant decorrelation and improved performance on a varoety of tasks using one of the input variables as the training signal for the secondary unit. Interestingly, the secondary task does not necessarily degrade performance on the primary task. Our studies, as well as those of Caruana (1995), show that extra tasks can facilitate learning time and generalization performance on an individual network. On the other hand, certain auxiliary tasks interfere with the primary task. We have found however, that even when the individual performance is degraded, committee performance is nevertheless enahnced (relative to a committee of single output networks) due to the magnitude of error decorrelation.

## 3 THE COMPETITIVE COMMITTEE

An alternative to using a stationary task per se, such as replicating an input variable or projecting onto principal components (as was done in Parmanto et al, 1994), is to use a signal that depends on the other networks, in such a manner that the functions computed by the secondary units are negatively correlated after training. This notion is reminiscent of competitive learning (Rumelhart and Zipser, 1986); that is, the functions computed by the secondary units will partition the stimulus space.

Thus, a Competitive Committee Machine (CCM) is defined as a committee of neural

network classifiers, each with two output units: a *primary* unit trained according to the classification task, and a *secondary* unit participating in a competitive process with secondary units of the other networks in the committee; let the outputs of network i be denoted $P_i$ and $S_i$, respectively (see Figure 1). The network weights are modified according to the following variant of the back propagation procedure.

When data item $\alpha$ from the training set is presented to the committee during training, with input vector $x^\alpha$ and known output classification value $y^\alpha$ (binary), the networks each process $x^\alpha$ simultaneously, and the P and S output units of each network respond. Each P-unit receives the identical training signal, $y^\alpha$, that corresponds to the input item; the training signal to the S-units is zero for all networks except the network with the greatest S-unit response among the committee; the maximum $S_i$ among the networks in the committee receives a training signal of 1, and the others receive a training signal of 0.

$$\delta_i^P = y^\alpha - P_i$$

$$\delta_i^S = \begin{cases} 1 - S_i & \text{if } S_i = \max_j S_j \\ -S_i & \text{otherwise} \end{cases}$$

where $\delta_i^P$ and $\delta_i^S$ are the errors attributed to the primary and secondary units respectively to adjust network weights with back propagation[1]. During the course of training, the S-unit's response is explicitly trained to become sensitive to a unique region (relative to the other networks' S-units) of the stimulus space. This training signal is different from typical "tasks" that are used to train neural networks in that it is not a static function of the input; instead, since it depends on the other networks in the committee, it has a dynamic quality.

## 4   RESULTS

Some experiments have been run using the sine wave classification task (Figure 2) of Geman and Bienenstock (1992).

Comparisons of CCM performance versus the baseline performance of a committee with a simple average over a range of architectures (as indicated by the number of hidden units) are favorable (Figure 3). Also, note that the improvement is primarily attributable to descreased correlation, since the average individual performance is not significantly affected.

Visualization of the response of the individual networks to the entire stimulus space gives a complete picture of how the networks generalize and shows the effect of the competition (Figure 4). For this particular data set, the classes are easily separated in the central region (note that all the networks do well here). But at the edges, there is much more variance in the networks trained with competitive secondary units (Figure 5).

## 5   DISCUSSION

Caruana (1995) has demontrated significant improvement on "target" classification tasks in individual networks by adding one or more supplementary output units trained to compute tasks related to the target task. The additional output unit added to each network

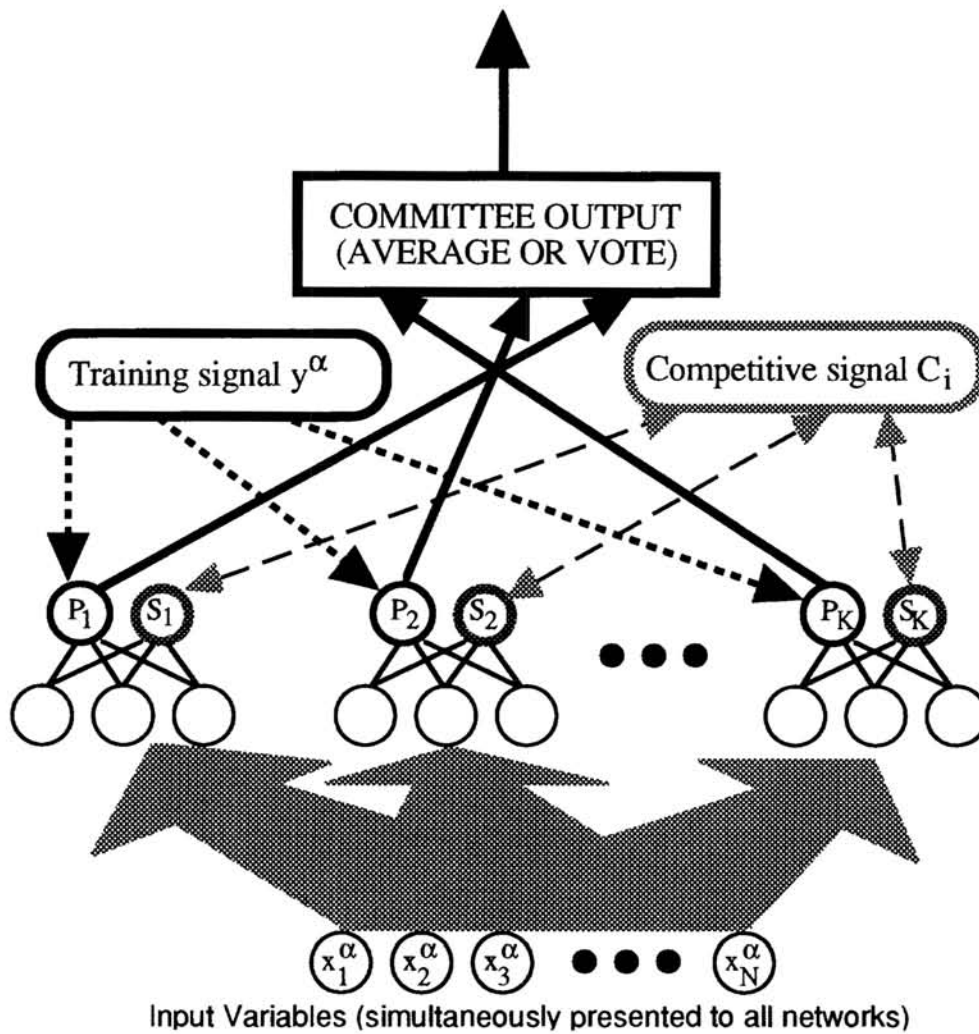

Input Variables (simultaneously presented to all networks)

Figure 1: *A Competitive Committee Machine.* Each of the K networks receives the same input and produces two outputs, P and S. The P responses of all the networks are compared to a common training signal to compute an error value for backpropagation (dark dashed arrows); the P responses are combined (by vote or by sum) to determine the committee response. The S-unit responses are compared with each other, with the "winner" (highest response) receiving a training signal of 1, and the others receiving a training signal of 0. Thus the training signal for network i is computed by comparing all S-unit responses, and then fed back to the S-units, hence the two-way arrows (gray).

in the CCM merges a variant of Rumelhart and Zipser's (1986) competitive learning procedure with backpropagation, to form a novel hybrid of a supervised training technique with an unsupervised method. The training signal delivered to the secondary unit under CCM is more *direct* than an arbitrary task, in that it is *defined* explicitly in terms of dissociating response properties.

Note that the training signals for the S-units differ from the P-unit training signals in two important respects:
1. Not static: The signal *depends on the S-unit responses from the other networks* and hence changes during the course of training.
2. Not uniform: It is not constant across the committee (whereas the P-unit training signal is.)

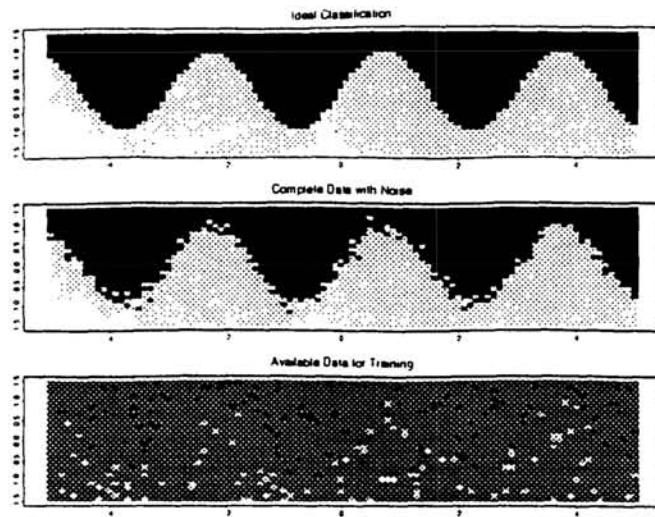

Figure 2. *A classification task*. Training data (bottom) is sampled from a classification task defined by a sinusoid (top) corrupted by noise (middle).

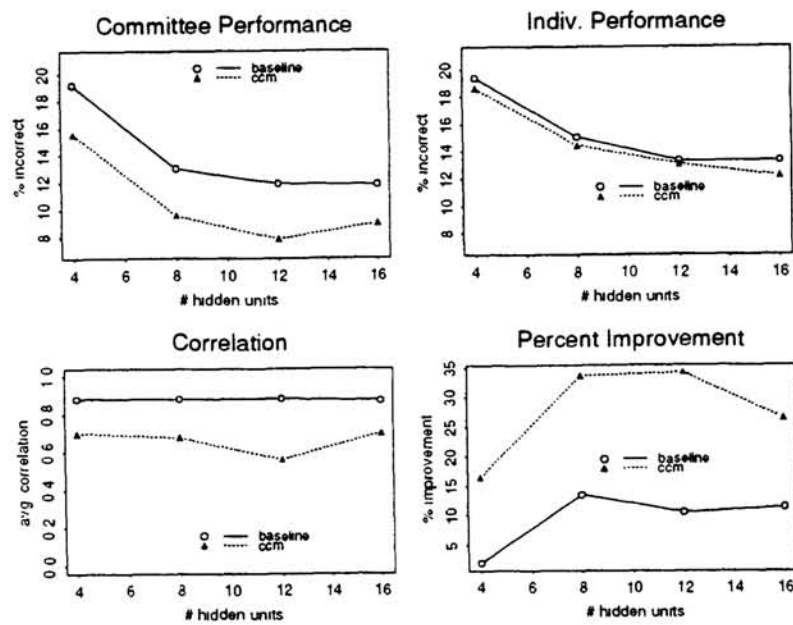

Figure 3. *Performance of CCM*. Committees of 5 networks were trained with competitive learning (CCM) and without (baseline). Each data point is an average over 5 simulations with different initial weights.

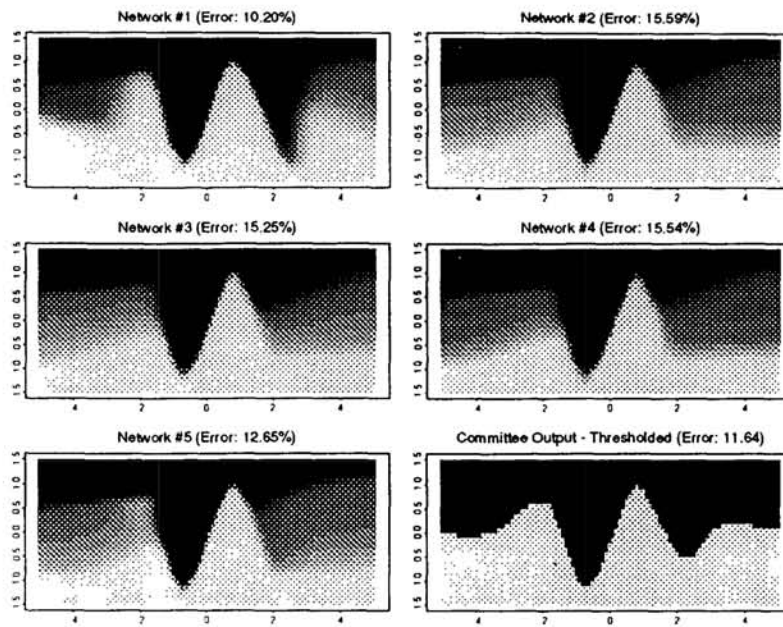

Figure 4. *Generalization plots for a committee.* The level of gray indicates the response for each network of a committee trained without competition. The panel on the lower right shows the (thresholded) committee output. The average pairwise correlation of the committee is 0.91.

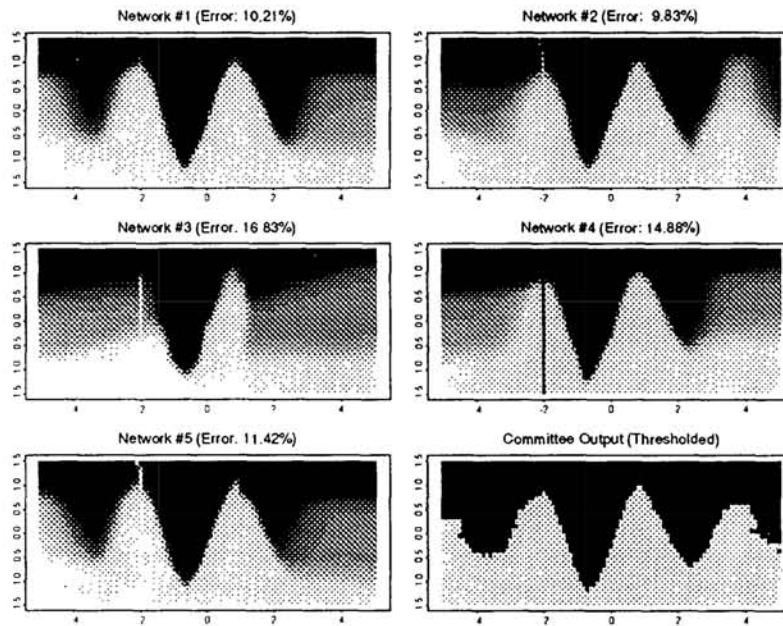

Figure 5. *Generalization plots for a CCM committee.* Comparison with Figure 4 shows much more variance among the committee at the edges. Note that the committee performs much better near the right and left ends of the stimulus space than does any individual network. This committee had an error rate of 8.11% (cf 11.64% in the baseline case).

The weighting of $\delta^S$ relative to $\delta^P$ is an important consideration; in the simulations above, the signal from the secondary unit was arbitrarily multiplied by a factor of 0.1. While we have not yet examined this systematically, it is assumed that this factor will modulate the tradeoff between degradation of the primary task and reduction of error correlation.

## Footnotes

[1]For notational convenience, the derivative factor sometimes included in the definition of $\delta$ is not included in this description of $\delta^P$ and $\delta^S$.

## References

Bates, J.M., and Granger, C.W. (1969) "The combination of forecasts," Operation Research Quarterly, 20(4), 451-468.

Breiman, L, (1992) "Stacked Regressions", TR 367, Dept. of Statistics, Univ. of Cal. Berkeley.

Caruana, R (1995) "Learning many related tasks at the same time with backpropagation," In: *Advances in Neural Information Processing Systems 7.* D. S. Touretsky, ed. Morgan Kaufmann.

Chauvin, Y. (1989) "A backpropagation algorithm with optimal use of hidden units." In Touretzky D., (ed.), *Advances in Neural Information Processing 1*, Denver, 1988, Morgan Kaufmann.

Geman, S., Bienenstock, E., and Doursat, R. (1992) "Neural networks and the bias/variance dilemma," *Neural Computation 4*, 1-58.

Hashem, S. (1994). Optimal Linear Combinations of Neural Networks., PhD Thesis, Purdue University.

Jacobs, R.A., Jordan, M.I., Nowlan, S.J., and Hinton, G.E. (1991) "Adaptive mixtures of local experts," *Neural Computation, 3*, 79-87

Le Cun, Y., Denker J. and Solla, S. (1990). Optimal Brain Damage. In D. Touretzky (Ed.) *Advances in Neural Information Processing Systems 2*, San Mateo: Morgan Kaufmann. 598-605.

Morgan, N. & Boulard, H. (1990) Generalization and parameter estimation in feedforward nets: some experiments. In D. Touretzky (Ed.) *Advances in   Neural Information Processing Systems 2* San Mateo: Morgan Kaufmann.

Parmanto, B., Munro, P.W., Doyle, H.R., Doria, C., Aldrighetti, L., Marino, I.R., Mitchel, S., and Fung, J.J. (1994) "Neural network classifier for hepatoma detection," *Proceedings of the World Congress of Neural Networks*

Parmanto, B., Munro, P.W., Doyle, H.R. (1996) "Improving committee diagnosis with resampling techniques," In: D. S. Touretzky, M. C. Mozer, M. E. Hasselmo, eds. *Advances in Neural Information Processing Systems 8.* MIT Press: Cambridge, MA.

Perrone, M.P. (1993) "Improving Regression Estimation: Averaging Methods for Variance Reduction with Extension to General Convex Measure Optimization," PhD Thesis, Department of Physics, Brown University.

Rumelhart. D.E and Zipser, D. (1986) "Feature discovery by competitive learning," In: Rumelhart, D.E.and McClelland, J.E. (Eds.), *Parallel Distributed Processing: Explorations in the Microstructure of Cognition.* MIT Press, Cambridge, MA.

Wolpert, D. (1992). Stacked generalization, *Neural Networks, 5*, 241-259.
